# A Local Learning Approach for Clustering

**Mingrui Wu, Bernhard Schölkopf**
Max Planck Institute for Biological Cybernetics
72076 Tübingen, Germany
{mingrui.wu, bernhard.schoelkopf}@tuebingen.mpg.de

## Abstract

We present a local learning approach for clustering. The basic idea is that a good clustering result should have the property that the cluster label of each data point can be well predicted based on its neighboring data and their cluster labels, using current supervised learning methods. An optimization problem is formulated such that its solution has the above property. Relaxation and eigen-decomposition are applied to solve this optimization problem. We also briefly investigate the parameter selection issue and provide a simple parameter selection method for the proposed algorithm. Experimental results are provided to validate the effectiveness of the proposed approach.

## 1 Introduction

In the multi-class clustering problem, we are given $n$ data points, $\mathbf{x}_1, \ldots, \mathbf{x}_n$, and a positive integer $c$. The goal is to partition the given data $\mathbf{x}_i$ ($1 \le i \le n$) into $c$ clusters, such that different clusters are in some sense "distinct" from each other. Here $\mathbf{x}_i \in \mathcal{X} \subseteq \mathbb{R}^d$ is the input data, $\mathcal{X}$ is the input space.

Clustering has been widely applied for data analysis tasks. It identifies groups of data, such that data in the same group are similar to each other, while data in different groups are dissimilar. Many clustering algorithms have been proposed, including the traditional k-means algorithm and the currently very popular spectral clustering approach [3, 10].

Recently the spectral clustering approach has attracted increasing attention due to its promising performance and easy implementation. In spectral clustering, the eigenvectors of a matrix are used to reveal the cluster structure in the data. In this paper, we propose a clustering method that also has this characteristic. But it is based on the local learning idea. Namely, the cluster label of each data point should be well estimated based on its neighboring data and their cluster labels, using current supervised learning methods. An optimization problem is formulated whose solution can satisfy this property. Relaxation and eigen-decomposition are applied to solve this problem. As will be seen later, the proposed algorithm is also easy to implement while it shows better performance than the spectral clustering approach in the experiments.

The local learning idea has already been successfully applied in supervised learning problems [1]. This motivates us to incorporate it into clustering, an important unsupervised learning problem. Adapting valuable supervised learning ideas for unsupervised learning problems can be fruitful. For example, in [9] the idea of large margin, which has proved effective in supervised learning, is applied to the clustering problem and good results are obtained.

The remaining of this paper is organized as follows. In section 2, we specify some notation that will be used in later sections. The details of our local learning based clustering algorithm are presented in section 3. Experimental results are then provided in section 4, where we also briefly investigate the parameter selection issue for the proposed algorithm. Finally we conclude the paper in the last section.

## 2 Notations

In the following, "neighboring points" or "neighbors" of $\mathbf{x}_i$ simply refers the nearest neighbors of $\mathbf{x}_i$ according to some distance metric.

| | |
|---|---|
| $n$ | the total number of data. |
| $c$ | the number of clusters to be obtained. |
| $\mathcal{C}_l$ | the set of points contained in the $l$-th cluster, $1 \leq l \leq c$. |
| $\mathcal{N}_i$ | the set of neighboring points of $\mathbf{x}_i$, $1 \leq i \leq n$, *not* including $\mathbf{x}_i$ itself. |
| $n_i$ | $|\mathcal{N}_i|$, i.e. the number of neighboring points of $\mathbf{x}_i$. |
| $Diag(\mathbf{M})$ | the diagonal matrix with the same size and the same diagonal elements as $\mathbf{M}$, where $\mathbf{M}$ is an arbitrary square matrix. |

## 3 Clustering via Local Learning

### 3.1 Local Learning in Supervised Learning

In supervised learning algorithms, a model is trained with all the labeled training data and is then used to predict the labels of unseen test data. These algorithms can be called global learning algorithms as the whole training dataset is used for training. In contrast, in local learning algorithms [1], for a given test data point, a model is built only with its neighboring training data, and then the label of the given test point is predicted by this locally learned model. It has been reported that local learning algorithms often outperform global ones [1] as the local models are trained only with the points that are related to the particular test data. And in [8], it is proposed that locality is a crucial parameter which can be used for capacity control, in addition to other capacity measures such as the VC dimension.

### 3.2 Representation of Clustering Results

The procedure of our clustering approach largely follows that of the clustering algorithms proposed in [2, 10]. We also use a *Partition Matrix* (PM) $\mathbf{P} = [p_{il}] \in \{0,1\}^{n \times c}$ to represent a clustering scheme. Namely $p_{il} = 1$ if $\mathbf{x}_i$ ($1 \leq i \leq n$) is assigned to cluster $\mathcal{C}_l$ ($1 \leq l \leq c$), otherwise $p_{il} = 0$. So in each row of $\mathbf{P}$, there is one and only one element that equals 1, all the others equal 0.

As in [2, 10], instead of computing the PM directly to cluster the given data, we compute a *Scaled Partition Matrix* (SPM) $\mathbf{F}$ defined by: $\mathbf{F} = \mathbf{P}(\mathbf{P}^\top \mathbf{P})^{-\frac{1}{2}}$. (The reason for this will be given later.)

As $\mathbf{P}^\top \mathbf{P}$ is diagonal, the $l$-th ($1 \leq l \leq c$) column of $\mathbf{F}$ is just the $l$-th column of $\mathbf{P}$ multiplied by $1/\sqrt{|\mathcal{C}_l|}$. Clearly we have

$$\mathbf{F}^\top \mathbf{F} = (\mathbf{P}^\top \mathbf{P})^{-\frac{1}{2}} \mathbf{P}^\top \mathbf{P} (\mathbf{P}^\top \mathbf{P})^{-\frac{1}{2}} = \mathbf{I} \tag{1}$$

where $\mathbf{I}$ is the unit matrix. Given a SPM $\mathbf{F}$, we can easily restore the corresponding PM $\mathbf{P}$ with a mapping $P(\cdot)$ defined as

$$\mathbf{P} = P(\mathbf{F}) = Diag(\mathbf{F}\mathbf{F}^\top)^{-\frac{1}{2}} \mathbf{F} \tag{2}$$

In the following, we will also express $\mathbf{F}$ as: $\mathbf{F} = [\mathbf{f}^1, \ldots, \mathbf{f}^c] \in \mathbb{R}^{n \times c}$, where $\mathbf{f}^l = [f_1^l, \ldots, f_n^l]^\top \in \mathbb{R}^n$, $1 \leq l \leq c$, is the $l$-th column of $\mathbf{F}$.

### 3.3 Basic Idea

The good performance of local learning methods indicates that *the label of a data point can be well estimated based on its neighbors.* Based on this, in order to find a good SPM $\mathbf{F}$ (or equivalently a good clustering result), we propose to solve the following optimization problem:

$$\min_{\mathbf{F} \in \mathbb{R}^{n \times c}} \quad \sum_{l=1}^{c} \sum_{i=1}^{n} (f_i^l - o_i^l(\mathbf{x}_i))^2 = \sum_{l=1}^{c} \left\| \mathbf{f}^l - \mathbf{o}^l \right\|^2 \tag{3}$$

$$\text{subject to} \quad \mathbf{F} \text{ is a scaled partition matrix} \tag{4}$$

where $o_i^l(\cdot)$ denotes the output function of a *Kernel Machine* (KM), trained with some supervised kernel learning algorithms [5], using the training data $\{(\mathbf{x}_j, f_j^l)\}_{\mathbf{x}_j \in \mathcal{N}_i}$, where $f_j^l$ is used as the label of $\mathbf{x}_j$ for training this KM. In (3), $\mathbf{o}^l = [o_1^l(\mathbf{x}_1), \ldots, o_n^l(\mathbf{x}_n)]^\top \in \mathbb{R}^n$. Details on how to compute $o_i^l(\mathbf{x}_i)$ will be given later. For the function $o_i^l(\cdot)$, the superscript $l$ indicates that it is for the $l$-th cluster, and the subscript $i$ means the KM is trained with the neighbors of $\mathbf{x}_i$. Hence apart from $\mathbf{x}_i$, the training data $\{(\mathbf{x}_j, f_j^l)\}_{\mathbf{x}_j \in \mathcal{N}_i}$ also influence the value of $o_i^l(\mathbf{x}_i)$. Note that $f_j^l$ ($\mathbf{x}_j \in \mathcal{N}_i$) are also variables of the problem (3)–(4).

To explain the idea behind problem (3)–(4), let us consider the following problem:

**Problem 1.** *For a data point $\mathbf{x}_i$ and a cluster $\mathcal{C}_l$, given the values of $f_j^l$ at $\mathbf{x}_j \in \mathcal{N}_i$, what should be the proper value of $f_i^l$ at $\mathbf{x}_i$?*

This problem can be solved by supervised learning. In particular, we can build a KM with the training data $\{(\mathbf{x}_j, f_j^l)\}_{\mathbf{x}_j \in \mathcal{N}_i}$. As mentioned before, let $o_i^l(\cdot)$ denote the output function of this locally learned KM, then the good performance of local learning methods mentioned above implies that $o_i^l(\mathbf{x}_i)$ is probably a good guess of $f_i^l$, or *the proper $f_i^l$ should be similar as $o_i^l(\mathbf{x}_i)$*.

Therefore, a good SPM $\mathbf{F}$ should have the following property: *For any $\mathbf{x}_i$ ($1 \leq i \leq n$) and any cluster $\mathcal{C}_l$ ($1 \leq l \leq c$), the value of $f_i^l$ can be well estimated based on the neighbors of $\mathbf{x}_i$. That is, $f_i^l$ should be similar to the output of the KM that is trained locally with the data $\{(\mathbf{x}_j, f_j^l)\}_{\mathbf{x}_j \in \mathcal{N}_i}$.* This suggests that in order to find a good SPM $\mathbf{F}$, we can solve the optimization problem (3)–(4).

We can also explain our approach intuitively as follows. A good clustering method will put the data into well separated clusters. This implies that it is easy to predict the cluster membership of a point based on its neighbors. If, on the other hand, a cluster is split in the middle, then there will be points at the boundary for which it is hard to predict which cluster they belong to. So minimizing the objective function (3) favors the clustering schemes that do not split the same group of data into different clusters.

Moreover, it is very difficult to construct local clustering algorithms in the same way as for supervised learning. In [1], a local learning algorithm is obtained by running a standard supervised algorithm on a local training set. This does not transfer to clustering. Rather than simply applying a given clustering algorithm locally and facing the difficulty to combine the local solution into a global one, problem (3)–(4) seeks a global solution with the property that locally for each point, its cluster assignment looks like the solution that we would obtain by local learning if we knew the cluster assignment of its neighbors.

## 3.4  Computing $o_i^l(\mathbf{x}_i)$

Having explained the basic idea, now we have to make the problem (3)–(4) more specific to build a concrete clustering algorithm. So we consider, based on $\mathbf{x}_i$ and $\{(\mathbf{x}_j, f_j^l)\}_{\mathbf{x}_j \in \mathcal{N}_i}$, how to compute $o_i^l(\mathbf{x}_i)$ with kernel learning algorithms. It is well known that applying many kernel learning algorithms on $\{(\mathbf{x}_j, f_j^l)\}_{\mathbf{x}_j \in \mathcal{N}_i}$ will result in a KM, according to which $o_i^l(\mathbf{x}_i)$ can be calculated as:

$$o_i^l(\mathbf{x}_i) = \sum_{\mathbf{x}_j \in \mathcal{N}_i} \beta_{ij}^l K(\mathbf{x}_i, \mathbf{x}_j) \tag{5}$$

where $K : \mathcal{X} \times \mathcal{X} \to \mathbb{R}$ is a positive definite kernel function [5], and $\beta_{ij}^l$ are the expansion coefficients. In general, any kernel learning algorithms can be applied to compute the coefficients $\beta_{ij}^l$. Here we choose the ones that make the problem (3)–(4) easy to solve. To this end, we adopt the *Kernel Ridge Regression* (KRR) algorithm [6], with which we can obtain an analytic expression of $o_i^l(\mathbf{x}_i)$ based on $\{(\mathbf{x}_j, f_j^l)\}_{\mathbf{x}_j \in \mathcal{N}_i}$. Thus for each $\mathbf{x}_i$, we need to solve the following KRR training problem:

$$\min_{\boldsymbol{\beta}_i^l \in \mathbb{R}^{n_i}} \lambda (\boldsymbol{\beta}_i^l)^\top \mathbf{K}_i \boldsymbol{\beta}_i^l + \left\| \mathbf{K}_i \boldsymbol{\beta}_i^l - \mathbf{f}_i^l \right\|^2 \tag{6}$$

where $\boldsymbol{\beta}_i^l \in \mathbb{R}^{n_i}$ is the vector of the expansion coefficients, i.e. $\boldsymbol{\beta}_i^l = [\beta_{ij}^l]^\top$ for $\mathbf{x}_j \in \mathcal{N}_i$, $\lambda > 0$ is the regularization parameter, $\mathbf{f}_i^l \in \mathbb{R}^{n_i}$ denotes the vector $\left[ f_j^l \right]^\top$ for $\mathbf{x}_j \in \mathcal{N}_i$, and $\mathbf{K}_i \in \mathbb{R}^{n_i \times n_i}$ is the kernel matrix over $\mathbf{x}_j \in \mathcal{N}_i$, namely $\mathbf{K}_i = [K(\mathbf{x}_u, \mathbf{x}_v)]$, for $\mathbf{x}_u, \mathbf{x}_v \in \mathcal{N}_i$.

Solving problem (6) leads to $\boldsymbol{\beta}_i^l = (\mathbf{K}_i + \lambda\mathbf{I})^{-1}\mathbf{f}_i^l$. Substituting it into (5), we have

$$o_i^l(\mathbf{x}_i) = \mathbf{k}_i^\top(\mathbf{K}_i + \lambda\mathbf{I})^{-1}\mathbf{f}_i^l \tag{7}$$

where $\mathbf{k}_i \in \mathbb{R}^{n_i}$ denotes the vector $[K(\mathbf{x}_i, \mathbf{x}_j)]^\top$ for $\mathbf{x}_j \in \mathcal{N}_i$. Equation (7) can be written as a linear equation:

$$o_i^l(\mathbf{x}_i) = \boldsymbol{\alpha}_i^\top\mathbf{f}_i^l \tag{8}$$

where $\boldsymbol{\alpha}_i \in \mathbb{R}^{n_i}$ is computed as

$$\boldsymbol{\alpha}_i^\top = \mathbf{k}_i^\top(\mathbf{K}_i + \lambda\mathbf{I})^{-1} \tag{9}$$

It can be seen that $\boldsymbol{\alpha}_i$ is independent of $\mathbf{f}_i^l$ and the cluster index $l$, and it is different for different $\mathbf{x}_i$. Note that $\mathbf{f}_i^l$ is a sub-vector of $\mathbf{f}^l$, so equation (8) can be written in a compact form as:

$$\mathbf{o}^l = \mathbf{A}\mathbf{f}^l \tag{10}$$

where $\mathbf{o}^l$ and $\mathbf{f}^l$ are the same as in (3), while the matrix $\mathbf{A} = [a_{ij}] \in \mathbb{R}^{n \times n}$ is constructed as follows: $\forall \mathbf{x}_i$ and $\mathbf{x}_j$, $1 \le i, j \le n$, if $\mathbf{x}_j \in \mathcal{N}_i$, then $a_{ij}$ equals the corresponding element of $\boldsymbol{\alpha}_i$ in (9), otherwise $a_{ij}$ equals 0. Similar as $\boldsymbol{\alpha}_i$, the matrix $\mathbf{A}$ is also independent of $\mathbf{f}^l$ and the cluster index $l$.

Substituting (10) into (3) results in a more specific optimization problem:

$$\min_{\mathbf{F} \in \mathbb{R}^{n \times c}} \quad \sum_{l=1}^{c} \left\| \mathbf{f}^l - \mathbf{A}\mathbf{f}^l \right\|^2 = \sum_{l=1}^{c}(\mathbf{f}^l)^\top\mathbf{T}\mathbf{f}^l = trace(\mathbf{F}^\top\mathbf{T}\mathbf{F}) \tag{11}$$

$$\text{subject to} \quad \mathbf{F} \text{ is a scaled partition matrix} \tag{12}$$

where

$$\mathbf{T} = (\mathbf{I} - \mathbf{A})^\top(\mathbf{I} - \mathbf{A}) \tag{13}$$

Thus, based on the KRR algorithm, we have transformed the objective function (3) into the quadratic function (11).

## 3.5  Relaxation

Following the method in [2, 10], we relax $\mathbf{F}$ into the continuous domain and combine the property (1) into the problem (11)–(12), so as to turn it into a tractable continuous optimization problem:

$$\min_{\mathbf{F} \in \mathbb{R}^{n \times c}} \quad trace(\mathbf{F}^\top\mathbf{T}\mathbf{F}) \tag{14}$$

$$\text{subject to} \quad \mathbf{F}^\top\mathbf{F} = \mathbf{I} \tag{15}$$

Let $\mathbf{F}^\star \in \mathbb{R}^{n \times c}$ denote the matrix whose columns consist of $c$ eigenvectors corresponding to the $c$ smallest eigenvalues of the symmetric matrix $\mathbf{T}$. Then it is known that the global optimum of the above problem is not unique, but a subspace spanned by the columns of $\mathbf{F}^\star$ through orthonormal matrices [10]:

$$\{\mathbf{F}^\star\mathbf{R} : \mathbf{R} \in \mathbb{R}^{c \times c}, \quad \mathbf{R}^\top\mathbf{R} = \mathbf{I}\} \tag{16}$$

Now we can see that working on the SPM $\mathbf{F}$ allows us to make use of the property (1) to construct a tractable continuous optimization problem (14)–(15), while working directly on the PM $\mathbf{P}$ does not have this advantage.

## 3.6  Discretization: Obtaining the Final Clustering Result

According to [10], to get the final clustering result, we need to find a true SPM $\mathbf{F}$ which is close to the subspace (16). To this end, we apply the mapping (2) on $\mathbf{F}^\star$ to obtain a matrix $\mathbf{P}^\star = P(\mathbf{F}^\star)$. It can be easily proved that for any orthogonal matrix $\mathbf{R} \in \mathbb{R}^{c \times c}$, we have $P(\mathbf{F}^\star\mathbf{R}) = \mathbf{P}^\star\mathbf{R}$. This equation implies that if there exists an orthogonal matrix $\mathbf{R}$ such that $\mathbf{F}^\star\mathbf{R}$ is close to a true SPM $\mathbf{F}$, then $\mathbf{P}^\star\mathbf{R}$ should also be near to the corresponding discrete PM $\mathbf{P}$. To find such an orthogonal matrix $\mathbf{R}$ and the discrete PM $\mathbf{P}$, we can solve the following optimization problem [10]:

$$\min_{\mathbf{P} \in \mathbb{R}^{n \times c}, \mathbf{R} \in \mathbb{R}^{c \times c}} \quad \left\| \mathbf{P} - \mathbf{P}^\star\mathbf{R} \right\|^2 \tag{17}$$

$$\text{subject to} \quad \mathbf{P} \in \{0, 1\}^{n \times c}, \quad \mathbf{P}\mathbf{1}_c = \mathbf{1}_n \tag{18}$$

$$\mathbf{R}^\top\mathbf{R} = \mathbf{I} \tag{19}$$

where $\mathbf{1}_c$ and $\mathbf{1}_n$ denote the $c$ dimensional and the $n$ dimensional vectors of all 1's respectively.

Details on how to find a local minimum of the above problem can be found in [10]. In [3], a method using k-means algorithm is proposed to find a discrete PM $\mathbf{P}$ based on $\mathbf{P}^\star$. In this paper, we adopt the approach in [10] to get the final clustering result.

### 3.7 Comparison with Spectral Clustering

Our *Local Learning based Clustering Algorithm* (LLCA) also uses the eigenvalues of a matrix ($\mathbf{T}$ in (13)) to reveal the cluster structure in the data, therefore it can be regarded as belonging to the category of spectral clustering approaches.

The matrix whose eigenvectors are used for clustering plays the key role in spectral clustering. In LLCA, this matrix is computed based on the local learning idea: a clustering result is obtained based on whether the label of each point can be well estimated base on its neighbors with a well established supervised learning algorithm. This is different from the graph partitioning based spectral clustering method. As will be seen later, LLCA and spectral clustering have quite different performance in the experiments.

LLCA needs one additional step: computing the matrix $\mathbf{T}$ in the objective function (14). The remaining steps, i.e. computing the eigenvectors of $\mathbf{T}$ and discretization (cf. section 3.6) are the same as in the spectral clustering approach. According to equation (13), to compute $\mathbf{T}$, we need to compute the matrix $\mathbf{A}$ in (10), which in turn requires calculating $\boldsymbol{\alpha}_i$ in (9) for each $\mathbf{x}_i$. We can see that this is very easy to implement and $\mathbf{A}$ can be computed with time complexity $O(\sum_{i=1}^{n} n_i^3)$.

In practice, just like in the spectral clustering method, the number of neighbors $n_i$ is usually set to a fixed small value $k$ for all $\mathbf{x}_i$ in LLCA. In this case, $\mathbf{A}$ can be computed efficiently with complexity $O(nk^3)$, which scales linearly with the number of data $n$. So in this case the main calculation is to obtain the eigenvectors of $\mathbf{T}$. Furthermore, according to (13), the eigenvectors of $\mathbf{T}$ are identical to the right singular vectors of $\mathbf{I} - \mathbf{A}$, which can be calculated efficiently because now $\mathbf{I} - \mathbf{A}$ is sparse, each row of which contains just $k + 1$ nonzero elements. Hence in this case, we do not need to compute $\mathbf{T}$ explicitly.

We conclude that LLCA is easy to implement, and in practice, the main computational load is to compute the eigenvectors of $\mathbf{T}$, therefore the LLCA and the spectral clustering approach have the same order of time complexity in most practical cases.[1]

## 4 Experimental Results

In this section, we empirically compare LLCA with the spectral clustering approach of [10] as well as with k-means clustering. For the last discretization step of LLCA (cf. section 3.6), we use the same code contained in the implementation of the spectral clustering algorithm, available at http://www.cis.upenn.edu/~jshi/software/.

### 4.1 Datasets

The following datasets are used in the experiments.

- USPS-3568: The examples of handwritten digits 3, 5, 6 and 8 from the USPS dataset.
- USPS-49: The examples of handwritten digits 4 and 9 from the USPS dataset.
- UMist: This dataset consists of face images of 20 different persons.
- UMist5: The data from the UMist dataset, belonging to class 4, 8, 12, 16 and 20.

- News4a: The text documents from the 20-newsgroup dataset, covering the topics in $rec.*$, which contains autos, motorcycles, baseball and hockey.
- News4b: The text documents from the 20-newsgroup dataset, covering the topics in $sci.*$, which contains crypt, electronics, med and space.

Further details of these datasets are provided in Table 1.

Table 1: Descriptions of the datasets used in the experiments. For each dataset, the number of data $n$, the data dimensionality $d$ and the number of classes $c$ are provided.

| Dataset | USPS-3568 | USPS-49 | UMist | UMist5 | News4a | News4b |
|---------|-----------|---------|-------|--------|--------|--------|
| $n$ | 3082 | 1673 | 575 | 140 | 3840 | 3874 |
| $d$ | 256 | 256 | 10304 | 10304 | 4989 | 5652 |
| $c$ | 4 | 2 | 20 | 5 | 4 | 4 |

In News4a and New4b, each document is represented by a feature vector, the elements of which are related to the frequency of occurrence of different words. For these two datasets, we extract a subset of each of them in the experiments by ignoring the words that occur in 10 or fewer documents and then removing the documents that have 10 or fewer words. This is why the data dimensionality are different in these two datasets, although both of them are from the 20-newsgroup dataset.

## 4.2 Performance Measure

In the experiments, we set the number of clusters equal to the number of classes $c$ for all the clustering algorithms. To evaluate their performance, we compare the clusters generated by these algorithms with the true classes by computing the following two performance measures.

### 4.2.1 Normalized Mutual Information

The *Normalized Mutual Information* (NMI) [7] is widely used for determining the quality of clusters. For two random variable $\mathbf{X}$ and $\mathbf{Y}$, the NMI is defined as [7]:

$$NMI(\mathbf{X}, \mathbf{Y}) = \frac{I(\mathbf{X}, \mathbf{Y})}{\sqrt{H(\mathbf{X})H(\mathbf{Y})}} \tag{20}$$

where $I(\mathbf{X}, \mathbf{Y})$ is the mutual information between $\mathbf{X}$ and $\mathbf{Y}$, while $H(\mathbf{X})$ and $H(\mathbf{Y})$ are the entropies of $\mathbf{X}$ and $\mathbf{Y}$ respectively. One can see that $NMI(\mathbf{X}, \mathbf{X}) = 1$, which is the maximal possible value of NMI. Given a clustering result, the NMI in (20) is estimated as [7]:

$$NMI = \frac{\sum_{l=1}^{c} \sum_{h=1}^{c} n_{l,h} log \left( \frac{n \cdot n_{l,h}}{n_l \hat{n}_h} \right)}{\sqrt{\left( \sum_{l=1}^{c} n_l log \frac{n_l}{n} \right) \left( \sum_{h=1}^{c} \hat{n}_h log \frac{\hat{n}_h}{n} \right)}} \tag{21}$$

where $n_l$ denotes the number of data contained in the cluster $\mathcal{C}_l$ ($1 \leq l \leq c$), $\hat{n}_h$ is the number of data belonging to the $h$-th class ($1 \leq h \leq c$), and $n_{l,h}$ denotes the number of data that are in the intersection between the cluster $\mathcal{C}_l$ and the $h$-th class. The value calculated in (21) is used as a performance measure for the given clustering result. The larger this value, the better the performance.

### 4.2.2 Clustering Error

Another performance measure is the *Clustering Error*. To compute it for a clustering result, we need to build a permutation mapping function $map(\cdot)$ that maps each cluster index to a true class label. The classification error based on $map(\cdot)$ can then be computed as:

$$err = 1 - \frac{\sum_{i=1}^{n} \delta(y_i, map(c_i))}{n}$$

where $y_i$ and $c_i$ are the true class label and the obtained cluster index of $\mathbf{x}_i$ respectively, $\delta(x, y)$ is the delta function that equals 1 if $x = y$ and equals 0 otherwise. The clustering error is defined as the minimal classification error among all possible permutation mappings. This optimal matching can be found with the Hungarian algorithm [4], which is devised for obtaining the maximal weighted matching of a bipartite graph.

### 4.3 Parameter Selection

In the spectral clustering algorithm, first a graph of $n$ nodes is constructed, each node of which corresponds to a data point, then the clustering problem is converted into a graph partition problem. In the experiments, for the spectral clustering algorithm, a weighted $k$-nearest neighbor graph is employed, where $k$ is a parameter searched over the grid: $k \in \{5, \ 10, \ 20, \ 40, \ 80\}$. On this graph, the edge weight between two connected data points is computed with a kernel function, for which the following two kernel functions are tried respectively in the experiments.

The cosine kernel:

$$K_1(\mathbf{x}_i, \mathbf{x}_j) = \frac{\mathbf{x}_i^\top \mathbf{x}_j}{\|\mathbf{x}_i\| \, \|\mathbf{x}_j\|} \tag{22}$$

and the Gaussian kernel:

$$K_2(\mathbf{x}_i, \mathbf{x}_j) = \exp(-\frac{1}{\gamma} \|\mathbf{x}_i - \mathbf{x}_j\|^2) \tag{23}$$

The parameter $\gamma$ in (23) is searched in: $\gamma \in \{\sigma_0^2/16, \ \sigma_0^2/8, \ \sigma_0^2/4, \ \sigma_0^2/2, \ \sigma_0^2, \ 2\sigma_0^2, \ 4\sigma_0^2, \ 8\sigma_0^2, \ 16\sigma_0^2\}$, where $\sigma_0$ is the mean norm of the given data $\mathbf{x}_i$, $1 \le i \le n$.

For LLCA, the cosine function (22) and the Gaussian function (23) are also adopted respectively as the kernel function in (5). The number of neighbors $n_i$ for all $\mathbf{x}_i$ is set to a single value $k$. The parameters $k$ and $\gamma$ are searched over the same grids as mentioned above. In LLCA, there is another parameter $\lambda$ (cf. (6)), which is selected from the grid: $\lambda \in \{0.1, \ 1, \ 1.5\}$.

Automatic parameter selection for unsupervised learning is still a difficult problem. We propose a simple parameter selection method for LLCA as follows. For a clustering result obtained with a set of parameters, which in our case consists of $k$ and $\lambda$ when the cosine kernel (22) is used, or $k$, $\gamma$ and $\lambda$ when the Gaussian kernel (23) is used, we compute its corresponding SPM $\mathbf{F}$ and then use the objective value (11) as the evaluation criteria. Namely, the clustering result corresponding to the smallest objective value is finally selected for LLCA.

For simplicity, on each dataset, we will just report the best result of spectral clustering. For LLCA, both the best result (LLCA1) and the one obtained with the above parameter selection method (LLCA2) will be provided. No parameter selection is needed for the k-means algorithm, since the number of clusters is given.

### 4.4 Numerical Results

Numerical results are summarized in Table 2. The results on News4a and News4b datasets show that different kernels may lead to dramatically different performance for both spectral clustering and LLCA. For spectral clustering, the results on USPS-3568 are also significantly different for different kernels. It can also be observed that different performance measures may result in different performance ranks of the clustering algorithms being investigated. This is reflected by the results on USPS-3568 when the cosine kernel is used and the results on News4b when the Gaussian kernel is used. Despite all these phenomena, we can still see from Table 2 that both LLCA1 and LLCA2 outperform the spectral clustering and the k-means algorithm in most cases.

We can also see that LLCA2 fails to find good parameters on News4a and News4b when the Gaussian kernel is used, while in the remaining cases, LLCA2 is either slightly worse than or identical to LLCA1. And analogous to LLCA1, LLCA2 also improves the results of the spectral clustering and the k-means algorithm on most datasets. This illustrates that our parameter selection method for LLCA can work well in many cases, and clearly it still needs improvement.

Finally, it can be seen that the k-means algorithm is worse than spectral clustering, except on USPS-3568 with respect to the clustering error criteria when the cosine kernel is used for spectral clustering. This corroborates the advantage of the popular spectral clustering approach over the traditional k-means algorithm.

## 5  Conclusion

We have proposed a local learning approach for clustering, where an optimization problem is formulated leading to a solution with the property that the label of each data point can be well estimated

Table 2: Clustering results. Both the normalized mutual information and the clustering error are provided. Two kernel functions (22) and (23) are tried for both spectral clustering and LLCA. On each dataset, the best result of the spectral clustering algorithm is reported (Spec-Clst). For LLCA, both the best result (LLCA1) and the one obtained with the parameter selection method described before (LLCA2) are provided. In each group, the best results are shown in boldface, the second best is in italics. Note that the results of k-means algorithm are independent of the kernel function.

|  |  | USPS-3568 | USPS-49 | UMist | UMist5 | News4a | News4b |
|---|---|---|---|---|---|---|---|
| NMI, cosine | Spec-Clst | *0.6575* | *0.3608* | 0.7483 | *0.8810* | *0.6468* | *0.5765* |
|  | LLCA1 | **0.8720** | **0.6241** | **0.8003** | **1** | **0.7587** | **0.7125** |
|  | LLCA2 | **0.8720** | **0.6241** | *0.7889* | **1** | **0.7587** | **0.7125** |
|  | k-means | 0.5202 | 0.2352 | 0.6479 | 0.7193 | 0.0800 | 0.0380 |
| NMI, Gaussian | Spec-Clst | 0.8245 | 0.4319 | *0.8099* | *0.8773* | **0.4039** | **0.1861** |
|  | LLCA1 | **0.8493** | **0.5980** | **0.8377** | **1** | *0.2642* | *0.1776* |
|  | LLCA2 | *0.8467* | *0.5493* | **0.8377** | **1** | 0.0296 | 0.0322 |
|  | k-means | 0.5202 | 0.2352 | 0.6479 | 0.7193 | 0.0800 | 0.0380 |
| Error (%), cosine | Spec-Clst | 32.93 | *16.56* | 46.26 | *9.29* | 28.26 | *21.73* |
|  | LLCA1 | **3.57** | **8.01** | **36.00** | **0** | **7.99** | **9.65** |
|  | LLCA2 | **3.57** | **8.01** | *38.43* | **0** | **7.99** | **9.65** |
|  | k-means | *22.16* | 22.30 | 56.35 | 36.43 | 70.62 | 74.08 |
| Error (%), Gaussian | Spec-Clst | 5.68 | 13.51 | 41.74 | *10.00* | **42.34** | *64.71* |
|  | LLCA1 | **4.61** | **8.43** | **33.91** | **0** | *47.24* | **53.25** |
|  | LLCA2 | *4.70* | *9.80* | *37.22* | **0** | 74.38 | 72.97 |
|  | k-means | 22.16 | 22.30 | 56.35 | 36.43 | 70.62 | 74.08 |

based on its neighbors. We have also provided a parameter selection method for the proposed clustering algorithm. Experiments show encouraging results. Future work may include improving the proposed parameter selection method and extending this work to other applications such as image segmentation.

## Footnotes

[1]Sometimes we are also interested in a special case: $n_i = n - 1$ for all $\mathbf{x}_i$, i.e. all the data are neighboring to each other. In this case, it can be proved that $\mathbf{T} = \mathbf{Q}^\top \mathbf{Q}$, where $\mathbf{Q} = (Diag(\mathbf{B}))^{-1}\mathbf{B}$ with $\mathbf{B} = \mathbf{I} - \mathbf{K}(\mathbf{K} + \lambda \mathbf{I})^{-1}$, where $\mathbf{K}$ is the kernel matrix over all the data points. So in this case $\mathbf{T}$ can be computed with time complexity $O(n^3)$. This is the same as computing the eigenvectors of the non-sparse matrix $\mathbf{T}$. Hence the order of the overall time complexity is not increased by the step of computing $\mathbf{T}$, and the above statements still hold.

# References

[1] L. Bottou and V. Vapnik. Local learning algorithms. *Neural Computation*, 4:888–900, 1992.

[2] P. K. Chan, M. D. F. Schlag, and J. Y. Zien. Spectral k-way ratio-cut partitioning and clustering. *IEEE Transactions on Computer-aided Design of Integrated Circuits and Systems*, 13:1088–1096, 1994.

[3] A. Y. Ng, M. I. Jordan, and Y. Weiss. On spectral clustering: analysis and an algorithm. In T. G. Dietterich, S. Becker, and Z. Ghahramani, editors, *Advances in Neural Information Processing Systems 14*, Cambridge, MA, 2002. MIT Press.

[4] C. H. Papadimitriou and K. Steiglitz. *Combinatorial Optimization: Algorithms and Complexity*. Dover, New York, 1998.

[5] B. Schölkopf and A. J. Smola. *Learning with Kernels*. The MIT Press, Cambridge, MA, 2002.

[6] J. Shawe-Taylor and N. Cristianini. *Kernel Methods for Pattern Analysis*. Cambridge University Press, Cambridge, UK, 2004.

[7] A. Strehl and J. Ghosh. Cluster ensembles – a knowledge reuse framework for combining multiple partitions. *Journal of Machine Learning Research*, 3:583–617, 2002.

[8] V. Vapnik. *The Nature of Statistical Learning Theory*. Springer Verlag, New York, 1995.

[9] L. Xu, J. Neufeld, B. Larson, and D. Schuurmans. Maximum margin clustering. In L. K. Saul, Y. Weiss, and L. Bottou, editors, *Advances in Neural Information Processing Systems 17*. MIT Press, Cambridge, MA, 2005.

[10] S. X. Yu and J. Shi. Multiclass spectral clustering. In L. D. Raedt and S. Wrobel, editors, *International Conference on Computer Vision*. ACM, 2003.
